# On Spectral Clustering:
# Analysis and an algorithm

**Andrew Y. Ng**
CS Division
U.C. Berkeley
*ang@cs.berkeley.edu*

**Michael I. Jordan**
CS Div. & Dept. of Stat.
U.C. Berkeley
*jordan@cs.berkeley.edu*

**Yair Weiss**
School of CS & Engr.
The Hebrew Univ.
*yweiss@cs.huji.ac.il*

## Abstract

Despite many empirical successes of *spectral clustering* methods—algorithms that cluster points using eigenvectors of matrices derived from the data—there are several unresolved issues. First, there are a wide variety of algorithms that use the eigenvectors in slightly different ways. Second, many of these algorithms have no proof that they will actually compute a reasonable clustering. In this paper, we present a simple spectral clustering algorithm that can be implemented using a few lines of Matlab. Using tools from matrix perturbation theory, we analyze the algorithm, and give conditions under which it can be expected to do well. We also show surprisingly good experimental results on a number of challenging clustering problems.

## 1 Introduction

The task of finding good clusters has been the focus of considerable research in machine learning and pattern recognition. For clustering points in $\mathbb{R}^n$—a main application focus of this paper—one standard approach is based on generative models, in which algorithms such as EM are used to learn a mixture density. These approaches suffer from several drawbacks. First, to use parametric density estimators, harsh simplifying assumptions usually need to be made (e.g., that the density of each cluster is Gaussian). Second, the log likelihood can have many local minima and therefore multiple restarts are required to find a good solution using iterative algorithms. Algorithms such as K-means have similar problems.

A promising alternative that has recently emerged in a number of fields is to use *spectral methods* for clustering. Here, one uses the top eigenvectors of a matrix derived from the distance between points. Such algorithms have been successfully used in many applications including computer vision and VLSI design [5, 1]. But despite their empirical successes, different authors still disagree on exactly which eigenvectors to use and how to derive clusters from them (see [11] for a review). Also, the analysis of these algorithms, which we briefly review below, has tended to focus on simplified algorithms that only use one eigenvector at a time.

One line of analysis makes the link to *spectral graph partitioning*, in which the sec-

ond eigenvector of a graph's Laplacian is used to define a semi-optimal cut. Here, the eigenvector is seen as a solving a relaxation of an NP-hard discrete graph partitioning problem [3], and it can be shown that cuts based on the second eigenvector give a guaranteed approximation to the optimal cut [9, 3]. This analysis can be extended to clustering by building a weighted graph in which nodes correspond to datapoints and edges are related to the distance between the points. Since the majority of analyses in spectral graph partitioning appear to deal with partitioning the graph into exactly two parts, these methods are then typically applied recursively to find $k$ clusters (e.g. [9]). Experimentally it has been observed that using more eigenvectors and directly computing a $k$ way partitioning is better (e.g. [5, 1]).

Here, we build upon the recent work of Weiss [11] and Meila and Shi [6], who analyzed algorithms that use $k$ eigenvectors simultaneously in simple settings. We propose a particular manner to use the $k$ eigenvectors simultaneously, and give conditions under which the algorithm can be expected to do well.

## 2  Algorithm

Given a set of points $S = \{s_1, \ldots, s_n\}$ in $\mathbb{R}^l$ that we want to cluster into $k$ subsets:

1. Form the affinity matrix $A \in \mathbb{R}^{n \times n}$ defined by $A_{ij} = \exp(-\|s_i - s_j\|^2 / 2\sigma^2)$ if $i \neq j$, and $A_{ii} = 0$.

2. Define $D$ to be the diagonal matrix whose $(i, i)$-element is the sum of $A$'s $i$-th row, and construct the matrix $L = D^{-1/2} A D^{-1/2}$.[1]

3. Find $x_1, x_2, \ldots, x_k$, the $k$ largest eigenvectors of $L$ (chosen to be orthogonal to each other in the case of repeated eigenvalues), and form the matrix $X = [x_1 x_2 \ldots x_k] \in \mathbb{R}^{n \times k}$ by stacking the eigenvectors in columns.

4. Form the matrix $Y$ from $X$ by renormalizing each of $X$'s rows to have unit length (i.e. $Y_{ij} = X_{ij} / (\sum_j X_{ij}^2)^{1/2}$).

5. Treating each row of $Y$ as a point in $\mathbb{R}^k$, cluster them into $k$ clusters via K-means or any other algorithm (that attempts to minimize distortion).

6. Finally, assign the original point $s_i$ to cluster $j$ if and only if row $i$ of the matrix $Y$ was assigned to cluster $j$.

Here, the scaling parameter $\sigma^2$ controls how rapidly the affinity $A_{ij}$ falls off with the distance between $s_i$ and $s_j$, and we will later describe a method for choosing it automatically. We also note that this is only one of a large family of possible algorithms, and later discuss some related methods (e.g., [6]).

At first sight, this algorithm seems to make little sense. Since we run K-means in step 5, why not just apply K-means directly to the data? Figure 1e shows an example. The natural clusters in $\mathbb{R}^2$ do not correspond to convex regions, and K-means run directly finds the unsatisfactory clustering in Figure 1i. But once we map the points to $\mathbb{R}^k$ ($Y$'s rows), they form tight clusters (Figure 1h) from which our method obtains the good clustering shown in Figure 1e. We note that the clusters in Figure 1h lie at $90°$ to each other relative to the origin (cf. [8]).

# 3 Analysis of algorithm

## 3.1 Informal discussion: The "ideal" case

To understand the algorithm, it is instructive to consider its behavior in the "ideal" case in which all points in different clusters are infinitely far apart. For the sake of discussion, suppose that $k = 3$, and that the three clusters of sizes $n_1$, $n_2$ and $n_3$ are $S_1$, $S_2$, and $S_3$ ($S = S_1 \cup S_2 \cup S_3$, $n = n_1 + n_2 + n_3$). To simplify our exposition, also assume that the points in $S = \{s_1, \dots, s_n\}$ are ordered according to which cluster they are in, so that the first $n_1$ points are in cluster $S_1$, the next $n_2$ in $S_2$, etc. We will also use "$j \in S_i$" as a shorthand for $s_j \in S_i$. Moving the clusters "infinitely" far apart corresponds to zeroing all the elements $A_{ij}$ corresponding to points $s_i$ and $s_j$ in different clusters. More precisely, define $\hat{A}_{ij} = 0$ if $x_i$ and $x_j$ are in different clusters, and $\hat{A}_{ij} = A_{ij}$ otherwise. Also let $\hat{L}$, $\hat{D}$, $\hat{X}$ and $\hat{Y}$ be defined as in the previous algorithm, but starting with $\hat{A}$ instead of $A$. Note that $\hat{A}$ and $\hat{L}$ are therefore block-diagonal:

$$\hat{A} = \begin{bmatrix} A^{(11)} & 0 & 0 \\ 0 & A^{(22)} & 0 \\ 0 & 0 & A^{(33)} \end{bmatrix}; \quad \hat{L} = \begin{bmatrix} \hat{L}^{(11)} & 0 & 0 \\ 0 & \hat{L}^{(22)} & 0 \\ 0 & 0 & \hat{L}^{(33)} \end{bmatrix} \qquad (1)$$

where we have adopted the convention of using parenthesized superscripts to index into subblocks of vectors/matrices, and $\hat{L}^{(ii)} = (\hat{D}^{(ii)})^{-1/2} A^{(ii)} (\hat{D}^{(ii)})^{-1/2}$. Here, $\hat{A}^{(ii)} = A^{(ii)} \in \mathbb{R}^{n_i \times n_i}$ is the matrix of "intra-cluster" affinities for cluster $i$. For future use, also define $\hat{d}^{(i)} \in \mathbb{R}^{n_i}$ to be the vector containing $\hat{D}^{(ii)}$'s diagonal elements, and $\hat{d} \in \mathbb{R}^n$ to contain $\hat{D}$'s diagonal elements.

To construct $\hat{X}$, we find $\hat{L}$'s first $k = 3$ eigenvectors. Since $\hat{L}$ is block diagonal, its eigenvalues and eigenvectors are the union of the eigenvalues and eigenvectors of its blocks (the latter padded appropriately with zeros). It is straightforward to show that $\hat{L}^{(ii)}$ has a strictly positive principal eigenvector $x_1^{(i)} \in \mathbb{R}^{n_i}$ with eigenvalue 1. Also, since $A_{jk}^{(ii)} > 0$ ($j \neq k$), the next eigenvalue is strictly less than 1. (See, e.g., [3]). Thus, stacking $\hat{L}$'s eigenvectors in columns to obtain $\hat{X}$, we have:

$$\hat{X} = \begin{bmatrix} x_1^{(1)} & \vec{0} & \vec{0} \\ \vec{0} & x_1^{(2)} & \vec{0} \\ \vec{0} & \vec{0} & x_1^{(3)} \end{bmatrix} \in \mathbb{R}^{n \times 3}. \qquad (2)$$

Actually, a subtlety needs to be addressed here. Since 1 is a repeated eigenvalue in $\hat{L}$, we could just as easily have picked any other 3 orthogonal vectors spanning the same subspace as $\hat{X}$'s columns, and defined them to be our first 3 eigenvectors. That is, $\hat{X}$ could have been replaced by $\hat{X}R$ for any orthogonal matrix $R \in \mathbb{R}^{3 \times 3}$ ($R^T R = RR^T = I$). Note that this immediately suggests that one use considerable caution in attempting to interpret the *individual* eigenvectors of $L$, as the choice of $\hat{X}$'s columns is arbitrary up to a rotation, and can easily change due to small perturbations to $A$ or even differences in the implementation of the eigensolvers. Instead, what we can reasonably hope to guarantee about the algorithm will be arrived at not by considering the (unstable) individual columns of $\hat{X}$, but instead the *subspace* spanned by the columns of $\hat{X}$, which can be considerably more stable. Next, when we renormalize each of $\hat{X}$'s rows to have unit length, we obtain:

$$\hat{Y} = \begin{bmatrix} \hat{Y}^{(1)} \\ \hat{Y}^{(2)} \\ \hat{Y}^{(3)} \end{bmatrix} = \begin{bmatrix} \vec{1} & \vec{0} & \vec{0} \\ \vec{0} & \vec{1} & \vec{0} \\ \vec{0} & \vec{0} & \vec{1} \end{bmatrix} R \qquad (3)$$

where we have used $\hat{Y}^{(i)} \in \mathbb{R}^{n_i \times k}$ to denote the $i$-th subblock of $\hat{Y}$. Letting $\hat{y}_j^{(i)}$

denote the $j$-th row of $\hat{Y}^{(i)}$, we therefore see that $\hat{y}_j^{(i)}$ is the $i$-th row of the orthogonal matrix $R$. This gives us the following proposition.

**Proposition 1** *Let $\hat{A}$'s off-diagonal blocks $\hat{A}^{(ij)}$, $i \neq j$, be zero. Also assume that each cluster $S_i$ is connected.[2] Then there exist $k$ orthogonal vectors $r_1, \ldots, r_k$ ($r_i^T r_j = 1$ if $i = j$, 0 otherwise) so that $\hat{Y}$'s rows satisfy*

$$\hat{y}_j^{(i)} = r_i \tag{4}$$

*for all $i = 1, \ldots, k$, $j = 1, \ldots, n_i$.*

In other words, there are $k$ mutually orthogonal points on the surface of the unit $k$-sphere around which $\hat{Y}$'s rows will cluster. Moreover, these clusters correspond exactly to the true clustering of the original data.

## 3.2 The general case

In the general case, $A$'s off-diagonal blocks are non-zero, but we still hope to recover guarantees similar to Proposition 1. Viewing $E = A - \hat{A}$ as a perturbation to the "ideal" $\hat{A}$ that results in $A = \hat{A} + E$, we ask: When can we expect the resulting rows of $Y$ to cluster similarly to the rows of $\hat{Y}$? Specifically, when will the eigenvectors of $L$, which we now view as a perturbed version of $\hat{L}$, be "close" to those of $\hat{L}$?

Matrix perturbation theory [10] indicates that the stability of the eigenvectors of a matrix is determined by the *eigengap*. More precisely, the *subspace* spanned by $\hat{L}$'s first 3 eigenvectors will be stable to small changes to $\hat{L}$ if and only if the eigengap $\delta = |\lambda_3 - \lambda_4|$, the difference between the 3rd and 4th eigenvalues of $\hat{L}$, is large. As discussed previously, the eigenvalues of $\hat{L}$ is the union of the eigenvalues of $\hat{L}^{(11)}$, $\hat{L}^{(22)}$, and $\hat{L}^{(33)}$, and $\lambda_3 = 1$. Letting $\lambda_j^{(i)}$ be the $j$-th largest eigenvalue of $\hat{L}^{(ii)}$, we therefore see that $\lambda_4 = \max_i \lambda_2^{(i)}$. Hence, the assumption that $|\lambda_3 - \lambda_4|$ be large is exactly the assumption that $\max_i \lambda_2^{(i)}$ be bounded away from 1.

**Assumption A1.** There exists $\delta > 0$ so that, for all $i = 1, \ldots, k$, $\lambda_2^{(i)} \leq 1 - \delta$.

Note that $\lambda_2^{(i)}$ depends only on $\hat{L}^{(ii)}$, which in turn depends only on $A^{(ii)} = \hat{A}^{(ii)}$, the matrix of intra-cluster similarities for cluster $S_i$. The assumption on $\lambda_2^{(i)}$ has a very natural interpretation in the context of clustering. Informally, it captures the idea that if we want an algorithm to find the clusters $S_1, S_2$ and $S_3$, then we require that each of these sets $S_i$ really look like a "tight" cluster. Consider an example in which $S_1 = S_{1.1} \cup S_{1.2}$, where $S_{1.1}$ and $S_{1.2}$ are themselves two well separated clusters. Then $S = S_{1.1} \cup S_{1.2} \cup S_2 \cup S_3$ looks like (at least) four clusters, and it would be unreasonable to expect an algorithm to correctly guess what partition of the four clusters into three subsets we had in mind.

This connection between the eigengap and the cohesiveness of the individual clusters can be formalized in a number of ways.

**Assumption A1.1.** Define the *Cheeger constant* [3] of the cluster $S_i$ to be

$$h(S_i) = \min_{\mathcal{I}} \frac{\sum_{j \in \mathcal{I}, k \notin \mathcal{I}} A_{jk}^{(ii)}}{\min\{\sum_{j \in \mathcal{I}} \hat{d}_j^{(i)}, \sum_{k \notin \mathcal{I}} \hat{d}_k^{(i)}\}}. \tag{5}$$

where the outer minimum is over all index subsets $\mathcal{I} \subseteq \{1, \ldots, n_i\}$. Assume that there exists $\delta > 0$ so that $(h(S_i))^2 / 2 \geq \delta$ for all $i$.

A standard result in spectral graph theory shows that Assumption A1.1 implies Assumption A1. Recall that $\hat{d}_j^{(i)} = \sum_k A_{jk}^{(ii)}$ characterizes how "well connected" or how "similar" point $j$ is to the other points in the same cluster. The term in the $\min_{\mathcal{I}}\{\cdot\}$ characterizes how well $(\mathcal{I}, \overline{\mathcal{I}})$ partitions $S_i$ into two subsets, and the minimum over $\mathcal{I}$ picks out the best such partition. Specifically, if there is a partition of $S_i$'s points so that the weight of the edges across the partition is small, and so that each of the partitions has moderately large "volume" (sum of $\hat{d}_j^{(i)}$'s), then the Cheeger constant will be small. Thus, the assumption that the Cheeger constants $h(S_i)$ be large is exactly that the clusters $S_i$ be hard to split into two subsets.

We can also relate the eigengap to the mixing time of a random walk (as in [6]) defined on the points of a cluster, in which the chance of transitioning from point $i$ to $j$ is proportional to $A_{ij}$, so that we tend to jump to nearby-points. Assumption A1 is equivalent to assuming that, for such a walk defined on the points of any one of the clusters $S_i$, the corresponding transition matrix has second eigenvalue at most $1 - \delta$. The mixing time of a random walk is governed by the second eigenvalue; thus, this assumption is exactly that the walks *mix rapidly*. Intuitively, this will be true for tight (or at least fairly "well connected") clusters, and untrue if a cluster consists of two well-separated sets of points so that the random walk takes a long time to transition from one half of the cluster to the other. Assumption A1 can also be related to the existence of multiple paths between any two points in the same cluster.

**Assumption A2.** There is some fixed $\epsilon_1 > 0$, so that for every $i_1, i_2 \in \{1, \dots, k\}$, $i_1 \neq i_2$, we have that

$$\sum_{j \in S_{i_1}} \sum_{k \in S_{i_2}} \frac{A_{jk}^2}{\hat{d}_j \hat{d}_k} \leq \epsilon_1. \tag{6}$$

To gain intuition about this, consider the case of two "dense" clusters $i_1$ and $i_2$ of size $\Omega(n)$ each. Since $\hat{d}_j$ measures how "connected" point $j$ is to other points in the same cluster, it will be $\hat{d}_j = \Omega(n)$ in this case, so the sum, which is over $O(n^2)$ terms, is in turn divided by $\hat{d}_j \hat{d}_k = \Omega(n^2)$. Thus, as long as the individual $A_{jk}$'s are small, the sum will also be small, and the assumption will hold with small $\epsilon_1$.

Whereas $\hat{d}_j$ measures how connected $s_j \in S_i$ is to the rest of $S_i$, $\sum_{k:k \notin S_i} A_{jk}$ measures how connected $s_j$ is to points in *other* clusters. The next assumption is that all points must be more connected to points in the same cluster than to points in other clusters; specifically, that the ratio between these two quantities be small.

**Assumption A3.** For some fixed $\epsilon_2 > 0$, for every $i = 1, \dots, k$, $j \in S_i$, we have:

$$\frac{\sum_{k:k \notin S_i} A_{jk}}{\hat{d}_j} \leq \epsilon_2 \left( \sum_{k,l \in S_i} \frac{A_{kl}^2}{\hat{d}_k \hat{d}_l} \right)^{-1/2} \tag{7}$$

For intuition about this assumption, again consider the case of densely connected clusters (as we did previously). Here, the quantity in parentheses on the right hand side is $O(1)$, so this becomes equivalent to demanding that the following ratio be small: $(\sum_{k:k \notin S_i} A_{jk})/\hat{d}_j = (\sum_{k:k \notin S_i} A_{jk})/(\sum_{k:k \in S_i} A_{jk}) = O(\epsilon_2)$.

**Assumption A4.** There is some constant $C > 0$ so that for every $i = 1, \dots, k$, $j = 1, \dots, n_i$, we have $\hat{d}_j^{(i)} \geq (\sum_{k=1}^{n_i} \hat{d}_k^{(i)})/(C n_i)$.

This last assumption is a fairly benign one that no points in a cluster be "too much less" connected than other points in the same cluster.

**Theorem 2** *Let assumptions A1, A2, A3 and A4 hold. Set $\epsilon = \sqrt{k(k-1)\epsilon_1 + k\epsilon_2^2}$.*

*If $\delta > (2 + \sqrt{2})\epsilon$, then there exist $k$ orthogonal vectors $r_1, \ldots, r_k$ ($r_i^T r_j = 1$ if $i = j$, 0 otherwise) so that $Y$'s rows satisfy*

$$\frac{1}{n} \sum_{i=1}^{k} \sum_{j=1}^{n_i} \|y_j^{(i)} - r_i\|_2^2 \leq 4C \left(4 + 2\sqrt{k}\right)^2 \frac{\epsilon^2}{(\delta - \sqrt{2}\epsilon)^2}. \qquad (8)$$

Thus, the rows of $Y$ will form tight clusters around $k$ well-separated points (at 90° from each other) on the surface of the $k$-sphere according to their "true" cluster $S_i$.

## 4 Experiments

To test our algorithm, we applied it to seven clustering problems. Note that whereas $\sigma^2$ was previously described as a human-specified parameter, the analysis also suggests a particularly simple way of choosing it automatically: For the right $\sigma^2$, Theorem 2 predicts that the rows of $Y$ will form $k$ "tight" clusters on the surface of the $k$-sphere. Thus, we simply search over $\sigma^2$, and pick the value that, after clustering $Y$'s rows, gives the tightest (smallest distortion) clusters. K-means in Step 5 of the algorithm was also inexpensively initialized using the prior knowledge that the clusters are about 90° apart.[3] The results of our algorithm are shown in Figure 1a-g. Giving the algorithm only the coordinates of the points and $k$, the different clusters found are shown in the Figure via the different symbols (and colors, where available). The results are surprisingly good: Even for clusters that do not form convex regions or that are not cleanly separated (such as in Figure 1g), the algorithm reliably finds clusterings consistent with what a human would have chosen.

We note that there are other, related algorithms that can give good results on a subset of these problems, but we are aware of no equally simple algorithm that can give results comparable to these. For example, we noted earlier how K-means easily fails when clusters do not correspond to convex regions (Figure 1i). Another alternative may be a simple "connected components" algorithm that, for a threshold $\tau$, draws an edge between points $s_i$ and $s_j$ whenever $\|s_i - s_j\|_2 \leq \tau$, and takes the resulting connected components to be the clusters. Here, $\tau$ is a parameter that can (say) be optimized to obtain the desired number of clusters $k$. The result of this algorithm on the `threecircles-joined` dataset with $k = 3$ is shown in Figure 1j. One of the "clusters" it found consists of a singleton point at $(1.5, 2)$. It is clear that this method is very non-robust.

We also compare our method to the algorithm of Meila and Shi [6] (see Figure 1k). Their method is similar to ours, except for the seemingly cosmetic difference that they normalize $A$'s rows to sum to 1 and use its eigenvectors instead of $L$'s, and do not renormalize the rows of $X$ to unit length. A refinement of our analysis suggests that this method might be susceptible to bad clusterings when the degree to which different clusters are connected ($\sum_j \hat{d}_j^{(i)}$) varies substantially across clusters.

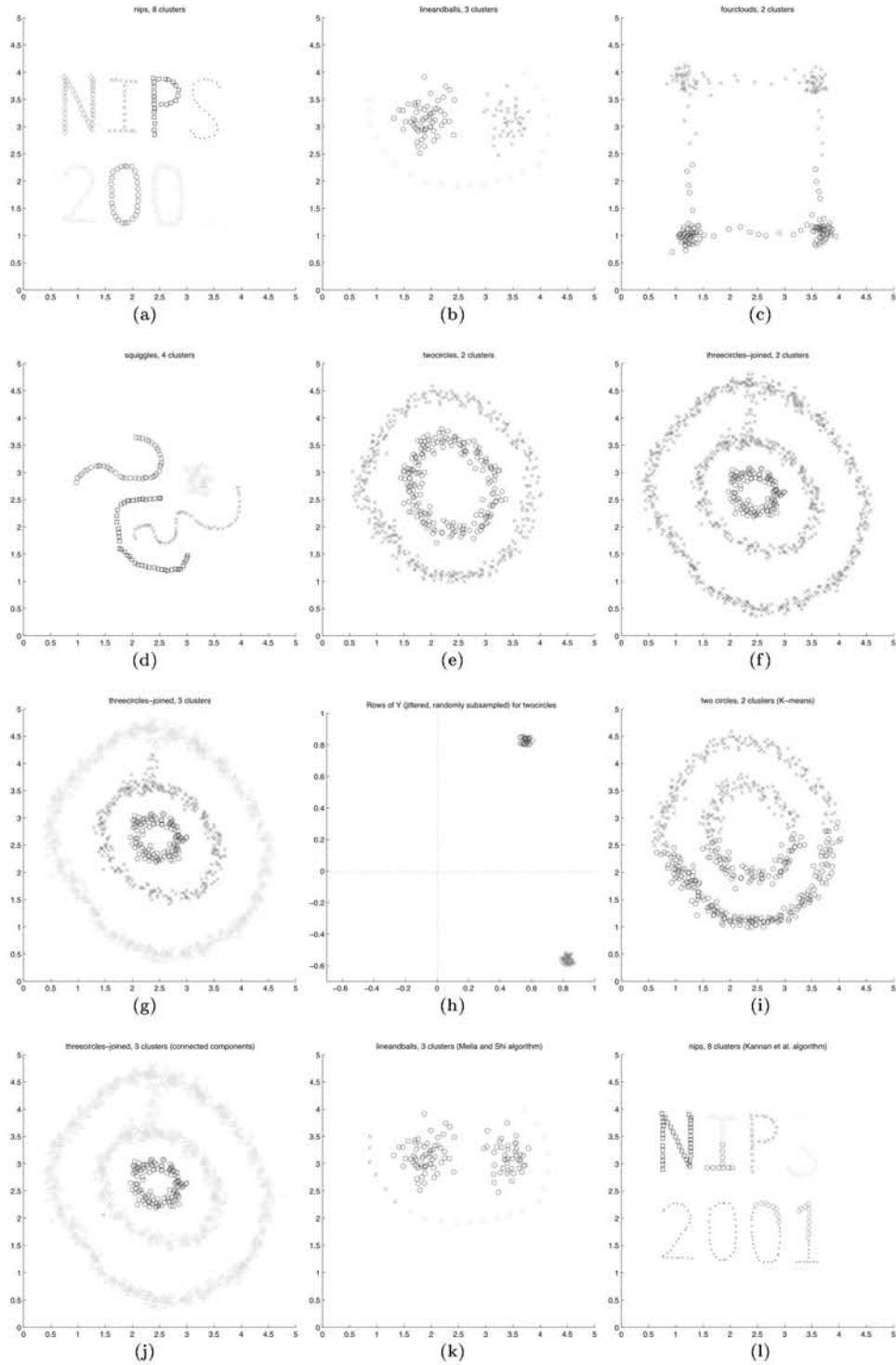

Figure 1: Clustering examples, with clusters indicated by different symbols (and colors, where available). (a-g) Results from our algorithm, where the only parameter varied across runs was $k$. (h) Rows of $Y$ (jittered, subsampled) for **twocircles** dataset. (i) K-means. (j) A "connected components" algorithm. (k) Meila and Shi algorithm. (l) Kannan et al. Spectral Algorithm I. (See text.)

# 5 Discussion

There are some intriguing similarities between spectral clustering methods and *Kernel PCA*, which has been empirically observed to perform clustering [7, 2]. The main difference between the first steps of our algorithm and Kernel PCA with a Gaussian kernel is the normalization of $A$ (to form $L$) and $X$. These normalizations do improve the performance of the algorithm, but it is also straightforward to extend our analysis to prove conditions under which Kernel PCA will indeed give clustering.

While different in detail, Kannan et al. [4] give an analysis of spectral clustering that also makes use of matrix perturbation theory, for the case of an affinity matrix with row sums equal to one. They also present a clustering algorithm based on $k$ singular vectors, one that differs from ours in that it identifies clusters with individual singular vectors. In our experiments, that algorithm very frequently gave poor results (e.g., Figure 1l).

### Acknowledgments

We thank Marina Meila for helpful conversations about this work. We also thank Alice Zheng for helpful comments. A. Ng is supported by a Microsoft Research fellowship. This work was also supported by a grant from Intel Corporation, NSF grant IIS-9988642, and ONR MURI N00014-00-1-0637.

## Footnotes

[1] Readers familiar with spectral graph theory [3] may be more familiar with the Laplacian $I - L$. But as replacing $L$ with $I - L$ would complicate our later discussion, and only changes the eigenvalues (from $\lambda_i$ to $1 - \lambda_i$) and not the eigenvectors, we instead use $L$.

[2] This condition is satisfied by $\hat{A}_{jk}^{(ii)} > 0$ ($j \neq k$), which is true in our case.

[3]Briefly, we let the first cluster centroid be a randomly chosen row of $Y$, and then repeatedly choose as the next centroid the row of $Y$ that is closest to being 90° from all the centroids (formally, from the worst-case centroid) already picked. The resulting K-means was run only once (no restarts) to give the results presented. K-means with the more conventional random initialization and a small number of restarts also gave identical results. In contrast, our implementation of Meila and Shi's algorithm used 2000 restarts.

# References

[1] C. Alpert, A. Kahng, and S. Yao. Spectral partitioning: The more eigenvectors, the better. *Discrete Applied Math*, 90:3–26, 1999.

[2] N. Christianini, J. Shawe-Taylor, and J. Kandola. Spectral kernel methods for clustering. In *Neural Information Processing Systems 14*, 2002.

[3] F. Chung. *Spectral Graph Theory*. Number 92 in CBMS Regional Conference Series in Mathematics. American Mathematical Society, 1997.

[4] R. Kannan, S. Vempala, and A. Vetta. On clusterings—good, bad and spectral. In *Proceedings of the 41st Annual Symposium on Foundations of Computer Science*, 2000.

[5] J. Malik, S. Belongie, T. Leung, and J. Shi. Contour and texture analysis for image segmentation. In *Perceptual Organization for Artificial Vision Systems*. Kluwer, 2000.

[6] M. Meila and J. Shi. Learning segmentation by random walks. In *Neural Information Processing Systems 13*, 2001.

[7] B. Schölkopf, A. Smola, and K.-R Müller. Nonlinear component analysis as a kernel eigenvalue problem. *Neural Computation*, 10:1299–1319, 1998.

[8] G. Scott and H. Longuet-Higgins. Feature grouping by relocalisation of eigenvectors of the proximity matrix. In *Proc. British Machine Vision Conference*, 1990.

[9] D. Spielman and S. Teng. Spectral partitioning works: Planar graphs and finite element meshes. In *Proceedings of the 37th Annual Symposium on Foundations of Computer Science*, 1996.

[10] G. W. Stewart and J.-G. Sun. *Matrix Perturbation Theory*. Academic Press, 1990.

[11] Y. Weiss. Segmentation using eigenvectors: A unifying view. In *International Conference on Computer Vision*, 1999.
